# Coordinate Transformation Learning of Hand Position Feedback Controller by Using Change of Position Error Norm

**Eimei Oyama***
Mechanical Eng. Lab.
Namiki 1-2, Tsukuba Science City
Ibaraki 305-8564 Japan

**Susumu Tachi**
The University of Tokyo
Hongo 7-3-1, Bunkyo-ku
Tokyo 113-0033 Japan

## Abstract

In order to grasp an object, we need to solve the inverse kinematics problem, i.e., the coordinate transformation from the visual coordinates to the joint angle vector coordinates of the arm. Although several models of coordinate transformation learning have been proposed, they suffer from a number of drawbacks. In human motion control, the learning of the hand position error feedback controller in the inverse kinematics solver is important. This paper proposes a novel model of the coordinate transformation learning of the human visual feedback controller that uses the change of the joint angle vector and the corresponding change of the square of the hand position error norm. The feasibility of the proposed model is illustrated using numerical simulations.

## 1 INTRODUCTION

The task of calculating every joint angle that would result in a specific hand position is called the inverse kinematics problem. An important topic in neuroscience is the study of the learning mechanisms involved in the human inverse kinematics solver.

We questioned five pediatricians about the motor function of infants suffering from serious upper limb disabilities. The doctors stated that the infants still were able to touch and stroke an object without hindrance. In one case, an infant without a thumb had a major kinematically influential surgical operation, transplanting an index finger as a thumb. After the operation, the child was able to learn how to use the index finger like a thumb [1]. In order to explain the human motor learning

capability, we believe that the coordinate transformation learning of the feedback controller is a necessary component.

Although a number of learning models of the inverse kinematics solver have been proposed, a definitive learning model has not yet been obtained. This is from the point of view of the structural complexity of the learning model and the biological plausibility of employed hypothesis. The *Direct Inverse Modeling* employed by many researchers [2] requires the complex switching of the input signal of the inverse model. When the hand position control is performed, the input of the inverse model is the desired hand position, velocity, or acceleration. When the inverse model learning is performed, the input is the observed hand position, velocity, or acceleration. Although the desired signal and the observed signal could coincide, the characteristics of the two signals are very different. Currently, no research has succeesfully modeled the switching system. Furthermore, that learning model is not "goal-directed"; i.e., there is no direct way to find an action that corresponds to a particular desired result. The *Forward and Inverse Modeling* proposed by Jordan [3] requires the back-propagation signal, a technique does not have a biological basis. That model also requires the complex switching of the desired output signal for the forward model. When the forward model learning is performed, the desired output is the observed hand position. When the inverse kinematics solver learning is performed, the desired output is the desired hand position. The *Feedback Error Learning* proposed by Kawato [4] requires a pre-existing accurate feedback controller.

It is necessary to obtain a learning model that possesses a number of characteristics: (1) it can explain the human learning function; (2) it has a simple structure; and (3) it is biologically plausible. This paper presents a learning model of coordinate transformation function of the hand position feedback controller. This model uses the joint angle vector change and the corresponding change of square of the hand position error norm.

## 2 BACKGROUND

### 2.1 Discrete Time First Order Model of Hand Position Controller

Let $\boldsymbol{\theta} \in \mathbf{R}^m$ be the joint angle vector and $\boldsymbol{x} \in \mathbf{R}^n$ be the hand position/orientation vector given by the vision system. The relationship between $\boldsymbol{x}$ and $\boldsymbol{\theta}$ is expressed as $\boldsymbol{x} = \boldsymbol{f}(\boldsymbol{\theta})$ where $\boldsymbol{f}$ is a $C^1$ class function. The Jacobian of the hand position vector is expressed as $\boldsymbol{J}(\boldsymbol{\theta}) = \partial \boldsymbol{f}(\boldsymbol{\theta})/\partial \boldsymbol{\theta}$. Let $\boldsymbol{x}_d$ be the desired hand position and $\boldsymbol{e} = \boldsymbol{x}_d - \boldsymbol{x} = \boldsymbol{x}_d - \boldsymbol{f}(\boldsymbol{\theta})$ be the hand position error vector. In this paper, an inverse kinematics problem is assumed to be a least squares minimization problem that calculates $\boldsymbol{\theta}$ in order to minimize the square of the hand position error norm $S(\boldsymbol{x}_d, \boldsymbol{\theta}) = |\boldsymbol{e}|^2/2 = |\boldsymbol{x}_d - \boldsymbol{f}(\boldsymbol{\theta})|^2/2$.

First, the feed-forward controller in the human inverse kinematics solver is disregarded and the following first order control system, consisting of a learning feedback controller, is considered:

$$\boldsymbol{\theta}(k + 1) = \boldsymbol{\theta}(k) + \boldsymbol{\Delta\theta}(k) \tag{1}$$

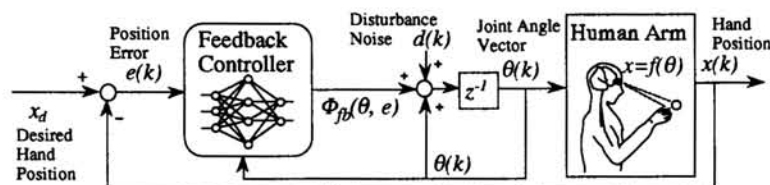

Figure 1: Configuration of 1-st Order Model of Hand Position Controller

$$\Delta\boldsymbol{\theta}(k) = \boldsymbol{\Phi}_{fb}(\boldsymbol{\theta}(k), e(k)) + \boldsymbol{d}(k) \tag{2}$$

$$e(k) = \boldsymbol{x}_d - \boldsymbol{f}(\boldsymbol{\theta}(k)) \tag{3}$$

where $\boldsymbol{d}(k)$ is assumed to be a disturbance noise from all components except the hand position control system. Figure 1 shows the configuration of the control system. In this figure, $z^{-1}$ is the operator that indicates a delay in the discrete time signal by a sampling interval of $\Delta t$. Although the human hand position control system includes higher order complex dynamics terms which are ignored in Equation (2), McRuer's experimental model of human compensation control suggests that the term that converts the hand position error to the hand velocity is a major term in the human control system [5]. We consider Equation (2) to be a good approximate model for the analysis of human coordinates transformation learning.

The learner $\boldsymbol{\Phi}_{fb}(\boldsymbol{\theta}, e) \in \mathbf{R}^m$, which provides the hand position error feedback, is modeled using the artificial neural network. In this paper, the hand position error feedback controller learning by observing output $\boldsymbol{x}(k)$ is considered without any prior knowledge of the function $\boldsymbol{f}(\boldsymbol{\theta})$.

## 2.2   Learning Model of the Neural Network

Let $\boldsymbol{\Phi}'_{fb}(\boldsymbol{\theta}, e)$ be the desired output of the learner $\boldsymbol{\Phi}_{fb}(\boldsymbol{\theta}, e)$. $\boldsymbol{\Phi}'_{fb}(\boldsymbol{\theta}, e)$ functions as a teacher for $\boldsymbol{\Phi}_{fb}(\boldsymbol{\theta}, e)$. Let $\boldsymbol{\Phi}^+_{fb}(\boldsymbol{\theta}, e)$ be the updated output of $\boldsymbol{\Phi}_{fb}(\boldsymbol{\theta}, e)$ by the learning. Let $E[t(\boldsymbol{\theta}, e)|\boldsymbol{\theta}, e]$ be the expected value of a scalar, a vector, or a matrix function $t(\boldsymbol{\theta}, e)$ when the input vector $(\boldsymbol{\theta}, e)$ is given. We assume that $\boldsymbol{\Phi}_{fb}(\boldsymbol{\theta}, e)$ is an ideal learner which is capable of realizing the mean of the desired output signal, completely. $\boldsymbol{\Phi}^+{}_{fb}(\boldsymbol{\theta}, e)$ can be expressed as follows:

$$\boldsymbol{\Phi}^+_{fb}(\boldsymbol{\theta}, e) \approx E[\boldsymbol{\Phi}'_{fb}(\boldsymbol{\theta}, e)|\boldsymbol{\theta}, e] = \boldsymbol{\Phi}_{fb}(\boldsymbol{\theta}, e) + E[\boldsymbol{\Delta\Phi}_{fb}(\boldsymbol{\theta}, e)|\boldsymbol{\theta}, e] \tag{4}$$

$$\boldsymbol{\Delta\Phi}_{fb}(\boldsymbol{\theta}, e) = \boldsymbol{\Phi}'_{fb}(\boldsymbol{\theta}, e) - \boldsymbol{\Phi}_{fb}(\boldsymbol{\theta}, e) \tag{5}$$

When the expected value of $\boldsymbol{\Delta\Phi}_{fb}(\boldsymbol{\theta}, e)$ is expressed as:

$$E[\boldsymbol{\Delta\Phi}_{fb}(\boldsymbol{\theta}, e)|\boldsymbol{\theta}, e] \approx \boldsymbol{G}_{fb}e - \boldsymbol{R}_{fb}\boldsymbol{\Phi}_{fb}(\boldsymbol{\theta}, e), \tag{6}$$

$\boldsymbol{R}_{fb} \in \mathbf{R}^{m \times m}$ is a positive definite matrix, and the inequality

$$\left|\frac{\partial \boldsymbol{\Phi}^+_{fb}(\boldsymbol{\theta}, e)}{\partial \boldsymbol{\Phi}_{fb}(\boldsymbol{\theta}, e)}\right| = \left|\frac{\partial(\boldsymbol{G}_{fb}e - (\boldsymbol{R}_{fb} - \boldsymbol{I})\boldsymbol{\Phi}_{fb}(\boldsymbol{\theta}, e))}{\partial \boldsymbol{\Phi}_{fb}(\boldsymbol{\theta}, e)}\right| < 1 \tag{7}$$

is satisfied, the final learning result can be expressed as:

$$\boldsymbol{\Phi}_{fb}(\boldsymbol{\theta}, e) \approx \boldsymbol{R}_{fb}^{-1}\boldsymbol{G}_{fb}e \tag{8}$$

by the iteration of the update of $\boldsymbol{\Phi}_{fb}(\boldsymbol{\theta}, e)$ expressed in Equation (4).

# 3   USE OF CHANGE OF POSITION ERROR NORM

## 3.1   A Novel Learning Model of Feedback Controller

The change of the square of the hand position error norm $\Delta S = S(\boldsymbol{x}_d, \boldsymbol{\theta} + \boldsymbol{\Delta\theta}) - S(\boldsymbol{x}_d, \boldsymbol{\theta})$ reflects whether or not the change of the joint angle vector $\boldsymbol{\Delta\theta}$ is in proper direction. The propose novel learning model can be expressed as follows:

$$\boldsymbol{\Phi}'_{fb}(\boldsymbol{\theta}, e) = -\alpha\Delta S\boldsymbol{\Delta\theta} \tag{9}$$

where $\alpha$ is a small positive real number. We now consider a large number of trials of Equation (2) with a large variety of initial status $\boldsymbol{\theta}(0)$ with learnings conducted at the point of the input space of the feedback controller $(\boldsymbol{\theta}, e) = (\boldsymbol{\theta}(k-1), e(k-1))$ at time $k$. $\Delta S$ and $\boldsymbol{\Delta\theta}$ can be calculated as follows.

$$\Delta S = S(k) - S(k-1) = \frac{1}{2}(|e(k)|^2 - |e(k-1)|^2) \tag{10}$$

$$\boldsymbol{\Delta\theta} = \boldsymbol{\Delta\theta}(k-1) \tag{11}$$

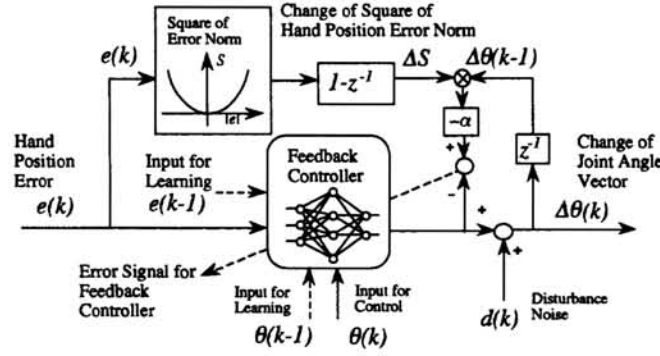

Figure 2: Configuration of Learning Model of Feedback Controller

Figure 2 shows the conceptual diagram of the proposed learning model.

Let $p(q|\theta, e)$ be the probability density function of a vector $q$ at at the point $(\theta, e)$ in the input space of $\Phi_{fb}(\theta, e)$. In order to simplify the analysis of the proposed learning model, $d(k)$ is assumed to satisfy the following equation:

$$p(d|\theta, e) = p(-d|\theta, e) \tag{12}$$

When $\Delta\theta$ is small enough, the result of the learning using Equation (9) can be expressed as:

$$\Phi_{fb}(\theta, e) \approx \alpha(\frac{\alpha}{2}R_\theta J^T(\theta)J(\theta) + I)^{-1}R_\theta J^T(\theta)e \tag{13}$$

$$R_\theta = E[\Delta\theta\Delta\theta^T|\theta, e] \tag{14}$$

where $J^T(\theta)e$ is a vector in the steepest descent direction of $S(x_d, \theta)$. When $d(k)$ is a non-zero vector, $R_\theta$ is a positive definite symmetric matrix and $(\frac{\alpha}{2}R_\theta J^T J + I)^{-1}$ is a positive definite matrix. When $\alpha$ is appropriate, $\Phi_{fb}(\theta, e)$ as expressed in Equation (13) can provide appropriate output error feedback control. The derivation of the above result will be illustrated in Section 3.2. A partially modified steepest descent direction can be obtained without using the forward model or the back-propagation signal, as Jordan's forward modeling [3].

Let $R_d$ be the covariance matrix of the disturbance noise $d(k)$. When $\alpha$ is infinitesimal, $R_\theta \approx R_d$ is established and an approximate solution $\Phi_{fb}(\theta, e) \approx \alpha R_d J^T(\theta)e$ is obtained.

### 3.2 Derivation of Learning Result

The change of the square of the hand position error norm $\Delta S(x_d, \theta)$ by $\Delta\theta$ can be determined as:

$$\Delta S(x_d, \theta) = \frac{\partial S(x_d, \theta)}{\partial \theta}\Delta\theta + \frac{1}{2}\Delta\theta^T H(x_d, \theta)\Delta\theta + O(\Delta\theta^3) \tag{15}$$

$$= -e^T(J(\theta) + \frac{1}{2}\frac{\partial J(\theta)}{\partial \theta} \otimes \Delta\theta)\Delta\theta + \frac{1}{2}\Delta\theta^T J^T(\theta)J(\theta)\Delta\theta + O(\Delta\theta^3)$$

where $\otimes$ is a 2-operand operator that indicates the Croneker's product. $H(x_d, \theta) \in R^{m \times m}$ is the Hessian of $S(x_d, \theta)$. $O(\Delta\theta^3)$ is the sum of third and higher order terms of $\Delta\theta$ in each equation. When $\Delta\theta$ is small enough, the following approximate equations are obtained:

$$\Delta x \approx J(\theta)\Delta\theta \approx J(\theta + \frac{1}{2}\Delta\theta)\Delta\theta \approx (J(\theta) + \frac{1}{2}\frac{\partial J(\theta)}{\partial \theta} \otimes \Delta\theta)\Delta\theta \tag{16}$$

Therefore, $\Delta S$ can be approximated as follows:

$$\Delta S \approx -e^T J(\theta)\Delta\theta + \frac{1}{2}|\Delta x|^2 \tag{17}$$

Since $e^T J \Delta\theta \Delta\theta = \Delta\theta \Delta\theta^T J^T e$ and $|\Delta x|^2 \Delta\theta = \Delta\theta \Delta\theta^T J^T J \Delta\theta$ are determined, $\Delta S \Delta\theta$ can be approximated as:

$$\Delta S \Delta\theta \approx -\Delta\theta \Delta\theta^T J^T(\theta)e + \frac{1}{2}\Delta\theta \Delta\theta^T J^T(\theta)J(\theta)\Delta\theta \qquad (18)$$

Considering $\Delta\theta_{nfb}$ defined as $\Delta\theta_{nfb} = \Delta\theta - \Phi_{fb}(\theta, e)$, the expected value of the product of $\Delta\theta$ and $\Delta S$ at the point $(\theta, e)$ in the input space of $\Phi_{fb}(\theta, e)$ can be approximated as follows:

$$E[\Delta S \Delta\theta | \theta, e] \approx -R_\theta J^T e + \frac{1}{2}R_\theta J^T J \Phi_{fb}(\theta, e) \qquad (19)$$
$$+ \frac{1}{2}E[\Delta\theta \Delta\theta^T J^T J \Delta\theta_{nfb}|\theta, e]$$

When the arm is controlled according to Equation (2), $\Delta\theta_{nfb}$ is the disturbance noise $d(k)$. Since $d(k)$ satisfies Equation (12), the following equation is established.

$$E[\Delta\theta \Delta\theta^T J^T J \Delta\theta_{nfb}|\theta, e] = 0 \qquad (20)$$

Therefore, the expected value of $\Delta\Phi_{fb}(\theta, e)$ can be expressed as;

$$E[\Delta\Phi_{fb}(\theta, e)|\theta, e] \approx \alpha R_\theta J^T e - (\frac{\alpha}{2}R_\theta J^T J + I)\Phi_{fb}(\theta, e) \qquad (21)$$

When $\alpha$ is small enough, the condition described in Equation (7) is established. The learning result expressed as Equation (13) is obtained as described in Section 2.2.

It should be noted that the learning algorithm expressed in Equation (9) is applicable not only to $S(x_d, \theta)$, but also to general penalty functions of hand position error norm $|e|$. The proposed learning model synthesizes a direction that decreases $S(x_d, \theta)$ by summing after weighting $\Delta\theta$ based on the increase or decrease of $S(x_d, \theta)$.

The feedback controller defined in Equation (13) requires a number of iterations to find a correct inverse kinematics solution, as the coordinates transformation function of the controller is incomplete. However, by using Kawato's feedback error learning [4], the second feedback controller; the feed-forward controller; or the inverse kinematics model that has a complete coordinate transformation function can be obtained as shown in Section 4.

## 4  TRACKING CONTROL SYSTEM LEARNING

In this section, we will consider the case where $x_d$ changes as $x_d(k)(k = 1, 2, \ldots)$. The hybrid controller that includes the learning feed-forward controller $\Phi_{ff}(\theta(k), \Delta x_d(k)) \in \mathbf{R}^m$ that transforms the change of the desired hand position $\Delta x_d(k) = x_d(k+1) - x_d(k)$ to the joint angle vector space is considered:

$$\Delta\theta(k) = \Phi_{ff}(\theta(k), \Delta x_d(k)) + \Phi_{fb}(\theta(k), e(k)) + d(k) \qquad (22)$$
$$e(k) = x_d(k) - x(k) \qquad (23)$$

The configuration of the hybrid controller is illustrated in Figure 3.
By using the modified change of the square of the error norm expressed as:

$$\Delta S = \frac{1}{2}(|x_d(k-1) - x(k)|^2 - |e(k-1)|^2) \qquad (24)$$

and $\Delta\theta(k)$ as defined in Equation (22), the feedback controller learning rule defined in Equation (9) is useful for the tracking control system. A sample holder for memorizing $x_d(k-1)$ is necessary for the calculation of $\Delta S$. When the distribution

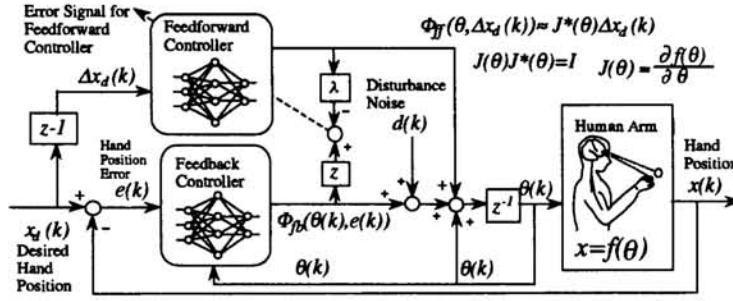

Figure 3: Configuration of Hybrid Controller

of $\Delta x_d(k)$ satisfies Equation (20), Equation (13) still holds. When $\Delta x_d(k)$ has no correlation with $d(k)$ and $\Delta x_d(k)$ satisfies $p(\Delta x_d|\theta, e) = p(-\Delta x_d|\theta, e)$, Equation (20) is approximately established after the feed-forward controller learning.

Using $\Delta\theta(k)$ defined in Equation (2) and $e(k)$ defined in Equation (23), $\Delta S$ defined in Equation (10) can be useful for the calculation of $\Phi'_{fb}(\theta, e)$. Although the learning calculation becomes simpler, the learning speed becomes much lower.

Let $\Phi'_{ff}(\theta(k), \Delta x_d(k))$ be the desired output of $\Phi_{ff}(\theta(k), \Delta x_d(k))$. According to Kawato's feedback error learning [4], we use $\Phi'_{ff}(\theta(k), \Delta x_d(k))$ expressed as:

$$\Phi'_{ff}(\theta(k), \Delta x_d(k)) = (1 - \lambda)\Phi_{ff}(\theta(k), \Delta x_d(k)) + \Phi_{fb}(\theta(k+1), e(k+1)) \quad (25)$$

where $\lambda$ is a small, positive, real number for stabilizing the learning process and ensuring that equation $\Phi_{ff}(\theta, 0) \approx 0$ holds. If $\lambda$ is small enough, the learning feed-forward controller will fulfill the equation:

$$J\Phi_{ff}(\theta, \Delta x_d) \approx \Delta x_d \quad (26)$$

## 5 NUMERICAL SIMULATION

Numerical simulation experiments were performed in order to evaluate the performance of the proposed model. The inverse kinematics of a 3 DOF arm moving on a 2 DOF plane were considered. The relationship between the joint angle vector $\theta = (\theta_1, \theta_2, \theta_3)^T$ and the hand position vector $x = (x, y)^T$ was defined as:

$$x = x_0 + L_1 \cos(\theta_1) + L_2 \cos(\theta_1 + \theta_2) + L_3 \cos(\theta_1 + \theta_2 + \theta_3) \quad (27)$$

$$y = y_0 + L_1 \sin(\theta_1) + L_2 \sin(\theta_1 + \theta_2) + L_3 \sin(\theta_1 + \theta_2 + \theta_3) \quad (28)$$

The range for $\theta_1$ was $(-30°, 120°)$; the range for $\theta_2$ was $(0°, 120°)$; and the range for $\theta_3$ was $(-75°, 75°)$. $L_1$ was 0.30 m, $L_2$ was 0.25 m and $L_3$ was 0.15 m.

Random straight lines were generated as desired trajectories for the hand. The tracking control trials expressed as Equation (22) with the learning of the feedback controller and the feed-forward controller were performed. The standard deviation of each component of $d$ was 0.01. Learnings based on Equations (9), (22), (24), and (25) were conducted 20 times in one tracking trial. 1,000 tracking trials were conducted to estimate the RMS(Root Mean Square) of $e(k)$.

In order to accelerate the learning, $\alpha$ in Equation (9) was modified as $\alpha = 0.5/(|\Delta x|^2 + 0.1|\Delta\theta|^2)$. $\lambda$ in Equation (25) was set to 0.001.

Two neural networks with 4 layers were used for the simulation. The first layer had 5 neurons and the forth layer had 3 neurons. The other layers had 15 neurons each. The first layer and the forth layer consisted of linear neurons. The initial values of weights of the neural networks were generated by using uniform random numbers. The back-propagation method without optimized learning coefficients was utilized for the learning.

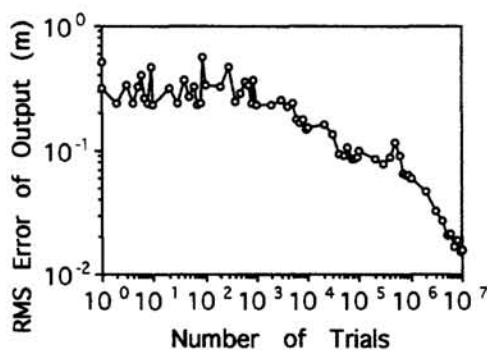
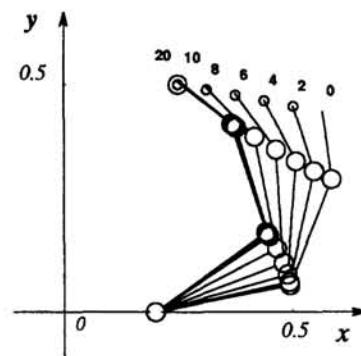

Figure 4: Learning Process of Controller    Figure 5: One Example of Tracking Control

Figure 4 shows the progress of the proposed learning model. It can be seen that the RMS error decreases and the precision of the solver becomes higher as the number of trials increases. The RMS error became $9.31 \times 10^{-3}$m after $2 \times 10^7$ learning trials. Figure 5 illustrates the hand position control by the inverse kinematics solver after $2 \times 10^7$ learning trials. The number near the end point of the arm indicates the value of $k$. The center of the small circle in Figure 5 indicates the desired hand position. The center of the large circle indicates the final desired hand position. Through learning, a precise inverse kinematics solver can be obtained. However, for RMS error to fall below 0.02, trials must be repeated more than $10^6$ times. In such cases, more efficient learner or a learning rule is necessary.

## 6 CONCLUSION

A learning model of coordinate transformation of the hand position feedback controller was proposed in this paper. Although the proposed learning model may take a long time to learn, it is capable of learning a correct inverse kinematics solver without using a forward model, a back-propagation signal, or a pre-existing feedback controller.

We believe that the slow learning speed can be improved by using neural networks that have a structure suitable for the coordinate transformation. A major limitation of the proposed model is the structure of the learning rule, since the learning rule requires the calculation of the product of the change of the error penalty function and the change of the joint angle vector. However, the existence of such structure in the nervous system is unknown. An advanced learning model which can be directly compared with the physiological and psychological experimental results is necessary.

## Footnotes

*Phone:+81-298-58-7298, Fax:+81-298-58-7201, e-mail:eimei@mel.go.jp

## References

[1] T. Ogino and S. Ishii, "Long-term Results after Pollicization for Congenital Hand Deformities," Hand Surgery, 2, 2,pp.79-85,1997

[2] F. H. Guenther and D. M. Barreca, " Neural models for flexible control of redundant systems," in P. Morasso and V. Sanguineti (Eds.), Self-organization, Computational Maps, and Motor Control. Amsterdam: Elsevier, pp.383-421,1997

[3] M. I. Jordan, "Supervised Learning and Systems with Excess Degrees of Freedom," COINS Technical Report,88-27,pp.1-41,1988

[4] M. Kawato, K. Furukawa and R. Suzuki, "A Hierarchical Neural-network Model for Control and Learning of Voluntary Movement," Biological Cybernetics, 57, pp.169-185, 1987

[5] D.T. McRuer and H. R. Jex, "A Review of Quasi-Linear Pilot Models," IEEE Trans. on Human Factors in Electronics, HFE-8, 3, pp.38-51, 1963
